# Variational Mixture of Gaussian Process Experts

**Chao Yuan and Claus Neubauer**
Siemens Corporate Research
Integrated Data Systems Department
755 College Road East, Princeton, NJ 08540
{chao.yuan,claus.neubauer}@siemens.com

## Abstract

Mixture of Gaussian processes models extended a single Gaussian process with ability of modeling multi-modal data and reduction of training complexity. Previous inference algorithms for these models are mostly based on Gibbs sampling, which can be very slow, particularly for large-scale data sets. We present a new generative mixture of experts model. Each expert is still a Gaussian process but is reformulated by a linear model. This breaks the dependency among training outputs and enables us to use a much faster variational Bayesian algorithm for training. Our gating network is more flexible than previous generative approaches as inputs for each expert are modeled by a Gaussian mixture model. The number of experts and number of Gaussian components for an expert are inferred automatically. A variety of tests show the advantages of our method.

## 1 Introduction

Despite of its widespread success in regression problems, Gaussian process (GP) has two limitations. First, it cannot handle data with multi-modality. Multi-modality can exist in the input dimension (e.g., non-stationarity), in the output dimension (given the same input, the output has multiple modes), or in a combination of both. Secondly, the cost of training is $\mathcal{O}(N^3)$, where $N$ is the size of the training set, which can be too expensive for large data sets. Mixture of GP experts models were proposed to tackle the above problems (Rasmussen & Ghahramani [1]; Meeds & Osindero [2]). Monte Carlo Markov Chain (MCMC) sampling methods (e.g., Gibbs sampling) are the standard approaches to train these models, which theoretically can achieve very accurate results. However, MCMC methods can be slow to converge and their convergence can be difficult to diagnose. It is thus important to explore alternatives.

In this paper, we propose a new generative mixture of Gaussian processes model for regression problems and apply variational Bayesian methods to train it. Each Gaussian process expert is described by a linear model, which breaks the dependency among training outputs and makes variational inference feasible. The distribution of inputs for each expert is modeled by a Gaussian mixture model (GMM). Thus, our gating network can handle missing inputs and is more flexible than single Gaussian-based gating models [2-4]. The number of experts and the number of components for each GMM are automatically inferred. Training using variational methods is much faster than using MCMC. The rest of this paper is organized as follows. Section 2 surveys the related work. Section 3 describes the proposed algorithm. We present test results in Section 4 and summarize this paper in Section 5.

## 2 Related work

Gaussian process is a powerful tool for regression problems (Rasmussen & Williams [5]). It elegantly models the dependency among data with a Gaussian distribution: $P(\mathbf{Y}) = \mathcal{N}(\mathbf{Y}|\mathbf{0}, \mathbf{K}+\sigma_n^2\mathbf{I})$,

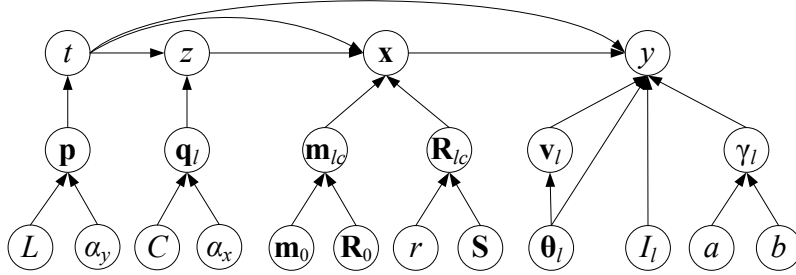

Figure 1: The graphical model representation for the proposed mixture of experts model. It consists of a hyperparameter set $\boldsymbol{\Theta} = \{L, \alpha_y, C, \alpha_x, \mathbf{m}_0, \mathbf{R}_0, r, \mathbf{S}, \boldsymbol{\theta}_{1:L}, \mathcal{I}_{1:L}, a, b\}$ and a parameter set $\boldsymbol{\Psi} = \{\mathbf{p}, \mathbf{q}_l, \mathbf{m}_{lc}, \mathbf{R}_{lc}, \mathbf{v}_l, \gamma_l \mid l = 1, 2, ..., L \text{ and } c = 1, 2, ..., C\}$. The local expert is a GP linear model to predict output $y$ from input $\mathbf{x}$; the gating network is a GMM for input $\mathbf{x}$. Data can be generated as follows. Step 1, determine hyperparameters $\boldsymbol{\Theta}$. Step 2, sample parameters $\boldsymbol{\Psi}$. Step 3, to sample one data point $\mathbf{x}$ and $y$, we sequentially sample expert indicator $t$, cluster indicator $z$, $\mathbf{x}$ and $y$. Step 3 is independently repeated until enough data points are generated.

where $\mathbf{Y} = \{y_{1:N}\}$ are $N$ training outputs and $\mathbf{I}$ is an identity matrix. We will use $y_{1:N}$ to denote $y_1, y_2, ..., y_N$. The kernel matrix $\mathbf{K}$ considered here consists of kernel functions between pairs of inputs $\mathbf{x}_i$ and $\mathbf{x}_j$: $K_{ij} = k(\mathbf{x}_i, \mathbf{x}_j) = \sigma_f^2 \exp(-\sum_{m=1}^{d} 1/(2\sigma_{gm}^2)(x_{im} - x_{jm})^2)$, where $d$ is the dimension of the input $\mathbf{x}$. The $d + 2$ hyperparameters $\sigma_n, \sigma_f, \sigma_{g1}, \sigma_{g2}, ..., \sigma_{gd}$ can be efficiently estimated from the data. However, Gaussian process has difficulties in modeling large-scale data and multi-modal data. The first issue was addressed by various sparse Gaussian processes [6-9, 16].

The mixture of experts (MoE) framework offers a natural solution for multi-modality problems (Jacobs *et al.* [10]). Early MoE work used linear experts [3, 4, 11, 12] and some of them were neatly trained via variational methods [4, 11, 12]. However, these methods cannot model nonlinear data sets well. Tresp [13] proposed a mixture of GPs model that can be trained fast using the EM algorithm. However, hyperparameters including the number of experts needed to be specified and the training complexity issue was not addressed. By introducing the Dirichlet process mixture (DPM) prior, infinite mixture of GPs models are able to infer the number of experts, both hyperparameters and parameters via Gibbs sampling [1, 2]. However, these models are trained by MCMC methods, which demand expensive training and testing time (as collected samples are usually combined to give predictive distributions). How to select samples and how many samples to be used are still challenging problems.

## 3 Algorithm description

Fig.1 shows the graphical model of the proposed mixture of experts. It consists of the local expert part and gating network part, which are covered in Sections 3.1 and 3.2, respectively. In Section 3.3, we describe how to perform variational inference of this model.

### 3.1 Local Gaussian process expert

A local Gaussian process expert is specified by the following linear model given the expert indicator $t = l$ (where $l = 1 : L$) and other related variables:

$$P(y|\mathbf{x}, t = l, \mathbf{v}_l, \boldsymbol{\theta}_l, \mathcal{I}_l, \gamma_l) = \mathcal{N}(y|\mathbf{v}_l^T \boldsymbol{\phi}_l(\mathbf{x}), \gamma_l^{-1}). \tag{1}$$

This linear model is symbolized by the inner product of the weight vector $\mathbf{v}_l$ and a nonlinear feature vector $\boldsymbol{\phi}_l(\mathbf{x})$. $\boldsymbol{\phi}_l(\mathbf{x})$ is a vector of kernel functions between a test input $\mathbf{x}$ and a subset of training inputs: $[k_l(\mathbf{x}, \mathbf{x}_{\mathcal{I}_{l1}}), k_l(\mathbf{x}, \mathbf{x}_{\mathcal{I}_{l2}}), ..., k_l(\mathbf{x}, \mathbf{x}_{\mathcal{I}_{lM}})]^T$. The *active set* $\mathcal{I}_l$ denotes the indices of selected $M$ training samples. How to select $\mathcal{I}_l$ will be addressed in Section 3.3; for now let us assume that we use the whole training set as the active set. $\mathbf{v}_l$ has a Gaussian distribution $\mathcal{N}(\mathbf{v}_l|\mathbf{0}, \mathbf{U}_l^{-1})$ with $\mathbf{0}$ mean and inverse covariance $\mathbf{U}_l$. $\mathbf{U}_l$ is set to $\mathbf{K}_l + \sigma_{hl}^2 \mathbf{I}$, where $\mathbf{K}_l$ is a $M \times M$ kernel matrix consisting of kernel functions between training samples in the active set. $\sigma_{hl}^2$ is needed to avoid singularity of $\mathbf{U}_l$. $\boldsymbol{\theta}_l = \{\sigma_{hl}, \sigma_{fl}, \sigma_{gl1}, \sigma_{gl2}, ..., \sigma_{gld}\}$ denotes the set of hyperparameters for this

linear model. Note that $\phi_l(\mathbf{x})$ depends on $\boldsymbol{\theta}_l$. $\gamma_l$ is the inverse variance of this linear model. The prior of $\gamma_l$ is set as a Gamma distribution: $\Gamma(\gamma_l|a,b) \propto b^a \gamma_l^{a-1} e^{-b\gamma_l}$ with hyperparameters $a$ and $b$.

It is easy to see that for each expert, $y$ is a Gaussian process defined on $\mathbf{x}$. Such a linear model was proposed by Silverman [14] and was used by sparse Gaussian process models [6, 8]. If we set $\sigma_{hl}^2 = 0$ and $\gamma_l = \frac{1}{\sigma_{nl}^2}$, the joint distribution of the training outputs $\mathbf{Y}$, assuming they are from the same expert $l$, can be proved to be $\mathcal{N}(\mathbf{Y}|\mathbf{0}, \mathbf{K}_l + \sigma_{nl}^2\mathbf{I})$. This has exactly the same form of a regular Gaussian process. However, the largest advantage of this linear model is that it breaks the dependency of $y_{1:N}$ once $t_{1:N}$ are given; i.e., $P(y_{1:N}|\mathbf{x}_{1:N}, t_{1:N}, \mathbf{v}_{1:L}, \boldsymbol{\theta}_{1:L}, \mathcal{I}_{1:L}, \gamma_{1:L}) = \prod_{n=1}^{N} P(y_n|\mathbf{x}_n, t_n = l, \mathbf{v}_l, \boldsymbol{\theta}_l, \mathcal{I}_l, \gamma_l)$. This makes the variational inference of the mixture of Gaussian processes feasible.

## 3.2 Gating network

A gating network determines which expert to use based on input $\mathbf{x}$. We consider a generative gating network, where expert indicator $t$ is generated by a categorical distribution $P(t = l) = p_l$. $\mathbf{p} = [p_1 \ p_2 \ ... \ p_L]$ is given a symmetric Dirichlet distribution $P(\mathbf{p}) = Dir(\mathbf{p}|\alpha_y/L, \alpha_y/L, ..., \alpha_y/L)$. Given expert indicator $t = l$, we assume that $\mathbf{x}$ follows a Gaussian mixture model (GMM) with $C$ components. Each component (cluster) is modeled by a Gaussian distribution $P(\mathbf{x}|t = l, z = c, \mathbf{m}_{lc}, \mathbf{R}_{lc}) = \mathcal{N}(\mathbf{x}|\mathbf{m}_{lc}, \mathbf{R}_{lc}^{-1})$. $z$ is the cluster indicator which has a categorical distribution $P(z = c|t = l, \mathbf{q}_l) = q_{lc}$. In addition, we give $\mathbf{m}_{lc}$ a Gaussian prior $\mathcal{N}(\mathbf{m}_{lc}|\mathbf{m}_0, \mathbf{R}_0^{-1})$, $\mathbf{R}_{lc}$ a Wishart prior $\mathcal{W}(\mathbf{R}_{lc}|r, \mathbf{S})$ and $\mathbf{q}_l$ a symmetric Dirichlet prior $Dir(\mathbf{q}_l|\alpha_x/C, \alpha_x/C, ..., \alpha_x/C)$.

In previous generative gating networks [2-4], the expert indicator also acts as the cluster indicator (or $t = z$) such that inputs for an expert can only have *one* Gaussian distribution. In comparison, our model is more flexible by modeling inputs $\mathbf{x}$ for each expert as a Gaussian mixture distribution. One can also put prior (e.g., inverse Gamma distribution) on $\alpha_x$ and $\alpha_y$ as done in [1, 2, 15]. In this paper we treat them as fixed hyperparameters.

## 3.3 Variational inference

**Variational EM algorithm** Given a set of training data $\mathbf{D} = \{(\mathbf{x}_n, y_n) \mid n = 1 : N\}$, the task of learning is to estimate unknown hyperparameters and infer posterior distribution of parameters. This problem is nicely addressed by the variational EM algorithm. The objective is to maximize $\log P(\mathbf{D}|\boldsymbol{\Theta})$ over hyperparameters $\boldsymbol{\Theta}$. Parameters $\boldsymbol{\Psi}$, expert indicators $\mathbf{T} = \{t_{1:N}\}$ and cluster indicators $\mathbf{Z} = \{z_{1:N}\}$ are treated as hidden variables, denoted by $\boldsymbol{\Omega} = \{\boldsymbol{\Psi}, \mathbf{T}, \mathbf{Z}\}$.

It is possible to estimate all hyperparameters via the EM algorithm. However, most of the hyperparameters are generic and are thus fixed as follows. $\mathbf{m}_0$ and $\mathbf{R}_0$ are set to be the mean and inverse covariance of the training inputs, respectively. We fix degree of freedom $r = d$ and the scale matrix $\mathbf{S} = 100\mathbf{I}$ for the Wishart distribution. $\alpha_x$, $\alpha_y$, $C$ and $L$ are all set to 10. Following Bishop & Svensén [12], we set $a = 0.01$ and $b = 0.0001$. Such settings give broad priors to the parameters and make our model sufficiently flexible. Our algorithm is not found to be sensitive to these generic hyperparameters. The only hyperparameters remain to be estimated are $\boldsymbol{\Theta} = \{\boldsymbol{\theta}_{1:L}, \mathcal{I}_{1:L}\}$. Note that these GP-related hyperparameters are problem specific and should not be assumed known.

In the E-step, based on the current estimates of $\boldsymbol{\Theta}$, posterior probability of hidden variables $P(\boldsymbol{\Omega}|\mathbf{D}, \boldsymbol{\Theta})$ is computed. Variational inference is involved in this step by approximating $P(\boldsymbol{\Omega}|\mathbf{D}, \boldsymbol{\Theta})$ with a factorized distribution

$$Q(\boldsymbol{\Omega}) = \prod_{l,c} Q(\mathbf{m}_{lc})Q(\mathbf{R}_{lc}) \prod_{l} Q(\mathbf{q}_l)Q(\mathbf{v}_l)Q(\gamma_l)Q(\mathbf{p}) \prod_{n} Q(t_n, z_n). \tag{2}$$

Each hidden variable has the same type of posterior distribution as its conjugate prior. To compute the distribution for a hidden variable $\omega_i$, we need to compute the posterior mean of $\log P(\mathbf{D}, \boldsymbol{\Omega}|\boldsymbol{\Theta})$ over all hidden variables except $\omega_i$: $\langle \log P(\mathbf{D}, \boldsymbol{\Omega}|\boldsymbol{\Theta}) \rangle_{\boldsymbol{\Omega}/\omega_i}$. The derivation is standard and is thus omitted.

Variational inference for each hidden variable takes linear time with respect to $N$, $C$ and $L$, because the factorized form of $P(\mathbf{D}, \boldsymbol{\Omega}|\boldsymbol{\Theta})$ leads to separation of hidden variables in $\log P(\mathbf{D}, \boldsymbol{\Omega}|\boldsymbol{\Theta})$. If we switch from our linear model to a regular Gaussian process, one will encounter a prohibitive

complexity of $\mathcal{O}(L^N)$ for integrating $\log P(y_{1:N}|\mathbf{x}_{1:N}, t_{1:N}, \mathbf{\Theta})$ over $t_{1:N}$. Also note that $C = L = 10$ represents the maximum number of clusters and experts. The actual number is usually smaller. During iteration, if a cluster $c$ for expert $l$ does not have a single training sample supporting it ($Q(t_n = l, z_n = c) > 0$), this cluster and its associated parameters $\mathbf{m}_{lc}$ and $\mathbf{R}_{lc}$ will be removed. Similarly, we remove an expert $l$ if no $Q(t_n = l) > 0$. These $C$ and $L$ choices are flexible enough for all our tests, but for more complicated data, larger values may be needed.

In the M-step, we search for $\mathbf{\Theta}$ which maximizes $\langle \log P(\mathbf{D}, \mathbf{\Omega}|\mathbf{\Theta}) \rangle_{\mathbf{\Omega}}$. We employ the conjugate gradient method to estimate $\boldsymbol{\theta}_{1:L}$ similarly to [5]. Both E-step and M-step are repeated until the algorithm converges. For better efficiency, we do not select the active sets $\mathcal{I}_{1:L}$ in each M-step; instead, we fix $\mathcal{I}_{1:L}$ during the EM algorithm and only update $\mathcal{I}_{1:L}$ once when the EM algorithm converges. The details are given after we introduce the algorithm initialization.

**Initialization** Without proper initialization, variational methods can be easily trapped into local optima. Consequently, using pure randomization methods, one cannot rely on a single result, but has to run the algorithm multiple times and then either pick the best result [12] or average the results [11]. We present a new initialization method that only needs the algorithm to run once. Our method is based on the assumption that the combined data including $\mathbf{x}$ and $y$ for an expert are usually distributed locally in the combined $d + 1$ dimensional space. Therefore, clustering methods such as $k$-mean can be used to cluster data, one cluster for one expert.

Experts are initialized incrementally as follows. First, all training data are used to train one expert. Secondly, we cluster all training data into two clusters and train one expert per cluster. We do this four times and collect a total of $L = 1 + 2 + 3 + 4 = 10$ experts. Different experts represent different local portions of training data in different scales. Although our assumption may not be true in some cases (e.g., one expert's data intersect with others), this initialization method does give us a meaningful starting point. In practice, we find it effective and reliable.

**Active set selection** We now address the problem of selecting active set $\mathcal{I}_l$ of size $M$ in defining the feature vector $\boldsymbol{\phi}_l$ for expert $l$. The posterior distribution $Q(\mathbf{v}_l)$ can be proved to be Gaussian with inverse covariance $\widetilde{\mathbf{U}}_l = \langle \gamma_l \rangle \sum_n T_{nl} \boldsymbol{\phi}_l(\mathbf{x}_n) \boldsymbol{\phi}_l(\mathbf{x}_n)^T + \mathbf{K}_l + \sigma_{hl}^2 \mathbf{I}$ and mean $\widetilde{\boldsymbol{\mu}}_l = \widetilde{\mathbf{U}}_l^{-1} \langle \gamma_l \rangle \sum_n T_{nl} y_n \boldsymbol{\phi}_l(\mathbf{x}_n)$. $T_{nl}$ is an abbreviation for $Q(t_n = l)$ and $\langle \gamma_l \rangle$ is the posterior mean of $\gamma_l$. Inverting $\widetilde{\mathbf{U}}_l$ has a complexity of $\mathcal{O}(M^3)$. Thus, for small data sets, the active set can be set to the full training set ($M = N$). But for large data sets, we have to select a subset with $M < N$.

The active set $\mathcal{I}_l$ is randomly initialized. With $\mathcal{I}_l$ fixed, we run the variational EM algorithm and obtain $Q(\mathbf{\Omega})$ and $\mathbf{\Theta}$. Now we want to improve our results by updating $\mathcal{I}_l$. Our method is inspired by the maximum a posteriori probability (MAP) used by sparse Gaussian processes [6, 8]. Specifically, the optimization target in our case is $\max_{\mathcal{I}_l, \mathbf{v}_l} P(\mathbf{v}_l|D) \approx Q(\mathbf{v}_l)$ with posterior distributions of other hidden variables fixed. The justification of this choice is that a good $\mathcal{I}_l$ should be strongly supported by data $\mathbf{D}$ such that $Q(\mathbf{v}_l)$ is highly peaked. Since $Q(\mathbf{v}_l)$ is Gaussian, $\mathbf{v}_l$ is always $\widetilde{\boldsymbol{\mu}}_l$ at the optimal point and thus this optimization is equivalent to maximizing the determinant of the inverse covariance

$$\max_{\mathcal{I}_l} |\widetilde{\mathbf{U}}_l| = |\langle \gamma_l \rangle \sum_n T_{nl} \boldsymbol{\phi}_l(\mathbf{x}_n) \boldsymbol{\phi}_l(\mathbf{x}_n)^T + \mathbf{K}_l + \sigma_{hl}^2 \mathbf{I}|. \tag{3}$$

Note that if $T_{nl}$ is one for all $n$, our method turns into a MAP-based sparse Gaussian process. However, even in that case, our criterion $\max_{\mathcal{I}_l, \mathbf{v}_l} P(\mathbf{v}_l|\mathbf{D})$ differs from $\max_{\mathcal{I}_l, \mathbf{v}_l} P(\mathbf{D}|\mathbf{v}_l)P(\mathbf{v}_l)$ derived in previous MAP-based work [6, 8]. First, the denominator $P(\mathbf{D})$ is ignored by previous methods, which actually depends on $\mathcal{I}_l$. Secondly, $|\mathbf{K}_l + \sigma_{hl}^2 \mathbf{I}|$ in $P(\mathbf{v}_l)$ is also ignored in previous methods. For these reasons, previous methods are not real MAP estimation but its approximations.

Looking for the global optimal active set with size $M$ is not feasible. Thus, similarly to many sparse Gaussian processes, we consider a greedy algorithm by adding one index to $\mathcal{I}_l$ each time. For a candidate index $i$, computing the new $\widetilde{\mathbf{U}}_l$ requires $\mathcal{O}(NM)$; incremental updating Cholesky factorization of $\widetilde{\mathbf{U}}_l$ requires $\mathcal{O}(M^2)$ and computing the new $|\widetilde{\mathbf{U}}_l|$ needs $\mathcal{O}(1)$. Therefore, checking one candidate $i$ takes $\mathcal{O}(NM)$. We consider selecting the best index from $\kappa = 100$ randomly selected candidates [6, 8], which makes the total time for adding one index $\mathcal{O}(\kappa NM)$. For adding all $M$ indices, the total time is $\mathcal{O}(\kappa NM^2)$. Such a complexity is comparable to that of [6], but higher than those of [7, 8]. Note that this time is needed for each of the $L$ experts.

In a summary, the variational EM algorithm with active set selection proceeds as follows. During initialization, training data are clustered and assigned to each expert by the $k$-mean clustering algorithm noted above; the data assigned to each expert is used for randomly selecting the active set and then training the linear model. During each iteration, we run variational EM to update parameters and hyperparameters; when the EM algorithm converges, we update the active set and $Q(\mathbf{v}_l)$ for each expert. Such an iteration is repeated until convergence.

It is also possible to define the feature vector $\phi_l(\mathbf{x})$ as $[k(\mathbf{x}, \overline{\mathbf{x}}_1), k(\mathbf{x}, \overline{\mathbf{x}}_2), ..., k(\mathbf{x}, \overline{\mathbf{x}}_M)]^T$, where each $\overline{\mathbf{x}}$ is a pseudo-input (Snelson & Ghahramani [9]). In this way, these pseudo-inputs $\overline{\mathbf{X}}$ can be viewed as hyperparameters and can be optimized in the same variational EM algorithm without resorting to a separate update for active sets as we do. This is theoretically more sound. However, it leads to a large number of hyperparameters to be optimized. Although overfitting may not be an issue, the authors cautioned that this method can be vulnerable to local optima.

**Predictive distribution** Once training is done, for a test input $\mathbf{x}^*$, its predictive distribution $P(y^*|\mathbf{x}^*, \mathbf{D}, \boldsymbol{\Theta})$ is evaluated as following:

$$P(y^*|\mathbf{x}^*, \mathbf{D}, \boldsymbol{\Theta}) = \int P(y^*|\mathbf{x}^*, \boldsymbol{\Omega}, \boldsymbol{\Theta})P(\boldsymbol{\Omega}|\mathbf{D}, \boldsymbol{\Theta})d\boldsymbol{\Omega} \approx \int P(y^*|\mathbf{x}^*, \boldsymbol{\Omega}, \boldsymbol{\Theta})Q(\boldsymbol{\Omega})d\boldsymbol{\Omega}$$

$$\approx P(y^*|\mathbf{x}^*, \langle \mathbf{p} \rangle, \{\langle \mathbf{q}_l \rangle\}, \{\langle \mathbf{m}_{lc} \rangle\}, \{\langle \mathbf{R}_{lc} \rangle\}, \{\langle \mathbf{v}_l \rangle\}, \{\langle \gamma_l \rangle\}, \{\boldsymbol{\theta}_l\}, \{\mathcal{I}_l\}). \quad (4)$$

The first approximation uses the results from the variational inference. Note that expert indicators $\mathbf{T}$ and cluster indicators $\mathbf{Z}$ are integrated out. Suppose that there are sufficient training data. Thus, the posterior distribution of all parameters are usually highly peaked. This leads to the second approximation, where the integral reduces to point evaluation at the posterior mean of each parameter. Eq.(4) can be easily computed using standard predictive algorithm for mixture of linear experts. See appendix for more details.

# 4 Test results

For all data sets, we normalize each dimension of data to zero mean and unit variance before using them for training. After training, to plot fitting results, we de-normalize data into their original scales.

**Artificial toy data** We consider the toy data set used by [2], which consists of four continuous functions covering input ranges $(0, 15)$, $(35, 60)$, $(45, 80)$ and $(80, 100)$, respectively. Different levels of noise (with standard deviations std $= 7, 7, 4$ and $2$) are added to different functions. This is a challenging multi-modality problem in both input and output dimensions. Fig.2 (left) shows 400 points generated by this toy model, each point with a equal probability 0.25 to be assigned to one of the four functions. Using these 400 points as training data, our method found two experts that fit the data nicely. Fig.2 (left) shows the results.

In general, expert one represents the last two functions while expert two represents the first two functions. One may desire to recover each function separately by an expert. However, note the fact that the first two functions have the same noise level (std $= 7$); so it is reasonable to use just one GP to model these two functions. In fact, we recovered a very close estimated std $= 1/\sqrt{\langle \gamma_2 \rangle} = 6.87$ for the second expert. The stds of the last two functions are also close (4 vs. 2), and are also similar to $1/\sqrt{\langle \gamma_1 \rangle} = 2.48$ of the first expert. Note that the GP for expert one appears to fit the data of the first function comparably well to that of expert two. However, the gating network does not support this: the means of the GMM for expert one does not cover the region of the first function.

Ref.[2] and our method performed similarly well in discovering different modalities in different input regions. We did not plot the mean of the predictive distribution as this data set has multiple modes in the output dimension. Our results were produced using an active set size $M = 60$. Larger active sets did not give appreciably better results.

**Motorcycle data** Our algorithm was also applied to the 2D motorcycle data set [14], which contains 133 data points with input-dependent noise as shown in Fig.2 (right). Our algorithm yielded two experts with the first expert modeling the majority of the points and the second expert only depicting the beginning part. The estimated stds of the two experts are 23.46 and 2.21, respectively. This appears to correctly represent different levels of noise present in different parts of the data.

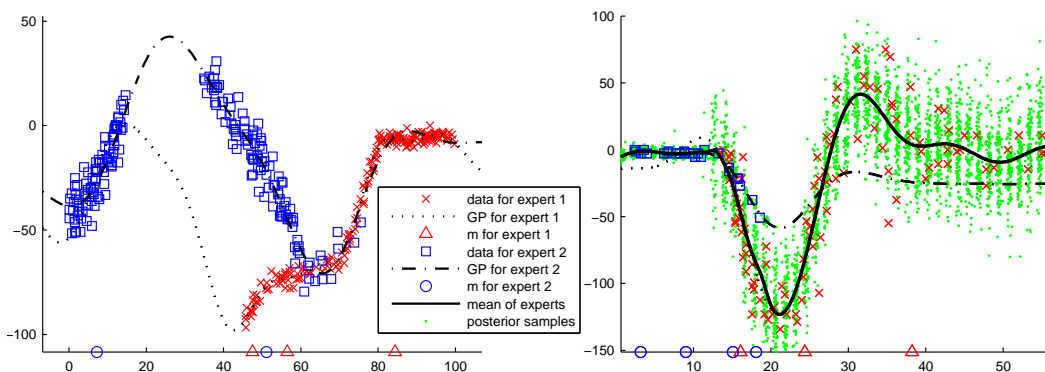

Figure 2: Test results for toy data (left) and motorcycle data (right). Each data point is assigned to an expert $l$ based on its posterior probability $Q(t_n = l)$ and is referred to as "data for expert $l$". The means of the GMM for each expert are also shown at the bottom as "m for expert $l$". In the right figure, the mean of the predictive distribution is shown as a solid line and samples drawn from the predictive distribution are shown as dots (100 samples for each of the 45 horizontal locations).

We also plot the mean of the predictive distribution (4) in Fig.2 (right). Our mean result compares favorably with other methods using medians of mixtures [1, 2]. In particular, our result is similar to that of [1] at input $\leq 30$. At input $> 35$, the result of [1] abruptly becomes flat while our result is smooth and appears to fit data better. The result of [2] is jagged, which may suggest using more Gibbs samples for smoother results. In terms of the full predictive (posterior) distribution (represented by samples in Fig.2 (right)), our results are better at input $\leq 40$ as more artifacts are produced by [1, 2] (especially between 15 and 25). However, our results have more artifacts at input $> 40$ because that region shares the same std $= 23.46$ as the other region where input is between 15 and 40. The active set size of our method is set to 40. Training using matlab 7 on a Pentium 2.4 GHz machine took 20 seconds, compared to one hour spent by Gibbs sampling method [1].

**Robot arm data** We consider the two-link robot arm data set used by [12]. Fig.3 (left) shows the kinematics of such a 2D robot. The joint angles are limited to the ranges $0.3 \leq \theta_1 \leq 1.2$ and $\pi/2 \leq \theta_2 \leq 3\pi/2$. Based on the forward kinematic equations (see [12]) the end point position $(x_1, x_2)$ has a unique solution given values of joint angles $(\theta_1, \theta_2)$. However, we are interested in the inverse kinematics problem: given the end point position, we want to estimate the joint angles. We randomly generated 2000 points based on the forward kinematics, with the first 1000 points for training and the remaining 1000 points for testing. Although noise can be added, we did not do so to make our results comparable to those of [12].

Since this problem involves predicting two correlated outputs at the same time, we used an independent set of local experts for each output but let these two outputs share the same gating network. This was easily adapted in our algorithm. Our algorithm found five experts vs. 16 experts used by [12]. The average number of GMM components is 3. We use residue plots [12] to present results (see Fig.3). Compared to that of [12], the first residue plot is much cleaner suggesting that our errors are much smaller. This is expected as we use more powerful GP experts vs. linear experts used by [12]. The second residue plot (not used in [12]) also gives clean result but is worse than the first plot. This is because the modality with the smaller posterior probability is more likely to be replaced by false positive modes. The active set size was set to 100. A larger size did not improve the results.

**DELVE data** We applied our algorithm to three widely used DELVE data sets: Boston, Kin-8nm and Pumadyn-32nm. These data sets appear to be single modal because impressive results were achieved by a single GP. The purpose of this test is to check how our algorithm (intended for multi-modality) handles single modality without knowing it. We followed the standard DELVE testing framework: for the Boston data, there are two tests each using 128 training examples; for both Kin-8nm and Pumadyn-32nm data, there are four tests, each using 1024 training examples.

Table 1 shows the standardised squared errors for the test. The scores from all previous methods are copied from Waterhouse [11]. We used the full training set as the active set. Reducing the active

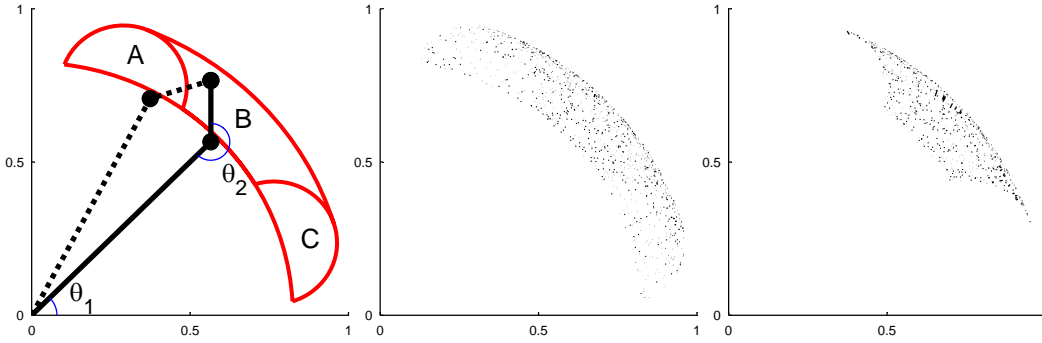

Figure 3: Test results for robot arm data set. Left: illustration of the robot kinematics (adapted from [12]). Our task is to estimate the joint angles $(\theta_1, \theta_2)$ based on the end point positions. In region B, there are two modalities for the same end point position. In regions A and C, there is only one modality. Middle: the first residue plot. For a test point, its predictive distribution is a Gaussian mixture. The mean of the Gaussian distribution with the highest probability was fed into the forward kinematics to obtain the estimated end point position. A line was drawn between the estimated and real end point positions; the length of the line indicates the magnitude of the error. The average line length (error) is a very small 0.00067 so many lines appear as dots. Right: the second residue plot using the mean of the Gaussian distribution with the second highest probability only for region B. The average line length is 0.001. Both residue plots are needed to check whether both modalities are detected correctly.

| Date sets | gp | mars | mlp | me | vmgp |
|---|---|---|---|---|---|
| Boston | $0.194 \pm 0.061$ | $\mathbf{0.157 \pm 0.009}$ | - | $0.159 \pm 0.023$ | $\mathbf{0.157 \pm 0.002}$ |
| Kin8nm | $0.116 \pm 0.006$ | $0.460 \pm 0.013$ | $\mathbf{0.094 \pm 0.013}$ | $0.182 \pm 0.020$ | $0.119 \pm 0.005$ |
| Pum32nm | $0.044 \pm 0.009$ | $0.061 \pm 0.003$ | $0.046 \pm 0.023$ | $0.701 \pm 0.079$ | $\mathbf{0.041 \pm 0.005}$ |

Table 1: Standardised squared errors of different methods on the DELVE data sets. Our method (vmgp) is compared with a single Gaussian process trained using a maximum a posteriori method (gp), a bagged version of MARS (mars), a multi-layer perceptron trained using hybrid MCMC (mlp) and a committee of mixtures of linear experts (me) [11].

set compromised the results, suggesting that for these high dimensional data sets, a large number of training examples are required; and for the present training sets, each training example carries information not represented by others. We started with ten experts and found an average of 2, 1 and 2.75 experts for these data sets, respectively. The average number of GMM components for these data sets are 8.5, 10 and 9.5, respectively, indicating that more GMM components are needed for modeling higher dimensional inputs. Our results are comparable to and sometimes better than those of previous methods.

Finally, to test how our active set selection algorithm performs, we conducted a standard test for sparse GPs: 7168 samples from Pumadyn-32nm were used for training and the remaining 1024 were for testing. The active set size $M$ was varied from 10 to 150. The error was 0.0569 when $M = 10$, but quickly reduced to 0.0225, the same as the benchmark error in [7], when $M = 25$. We rapidly achieved 0.0196 at $M = 50$ and the error did not decrease after that. This result is better than that of [7] and comparable to the best result of [9].

## 5   Conclusions

We present a new mixture of Gaussian processes model and apply variational Bayesian method to train it. The proposed algorithm nicely addresses data multi-modality and training complexity issues of a single Gaussian process. Our method achieved comparable results to previous MCMC-based models on several 2D data sets. One future direction is to compare all algorithms using high dimensional data so we can draw more meaningful conclusions. However, one clear advantage of

our method is that training is much faster. This makes our method more suitable for many real-world applications where speed is critical.

Our active set selection method works well on the Pumadyn-32nm data set. But this test was done in the context of mixture of GPs. To make a fair comparison to other sparse GPs, we can set $L = 1$ and also try more data sets. It is worthy noting that in the current implementation, the active set size $M$ is fixed for all experts. This can be improved by using a smaller $M$ for an expert with a smaller number of supporting training samples.

**Acknowledgments**

Thanks to Carl Rasmussen and Christopher Williams for sharing the GPML matlab package.

**Appendix**

Eq.(4) can be expressed as a weighted sum of all experts, where hyperparameters and parameters are omitted:

$$P(y^*|\mathbf{x}^*) = \sum_l \sum_c P(t^* = l, z^* = c|\mathbf{x}^*)P(y^*|\mathbf{x}^*, t^* = l). \qquad \text{(A-1)}$$

The first term in (A-1) is the posterior probability for expert $t^* = l$ and it is the sum of

$$P(t^* = l, z^* = c|\mathbf{x}^*) = \frac{P(\mathbf{x}^*|t^* = l, z^* = c)P(t^* = l, z^* = c)}{\sum_{l'} \sum_{c'} P(\mathbf{x}^*|t^* = l', z^* = c')P(t^* = l', z^* = c')}, \qquad \text{(A-2)}$$

where $P(t^* = l, z^* = c) = \langle p_l \rangle \langle q_{lc} \rangle$. The second term in (A-1) is the predictive probability for $y^*$ given expert $l$, which is Gaussian.

# References

[1] C. E. Rasmussen and Z. Ghahramani. Infinite mixtures of Gaussian process experts. In *Advances in Neural Information Processing Systems 14*. MIT Press, 2002.

[2] E. Meeds and S. Osindero. An alternative infinite mixture of Gaussian process experts. In *Advances in Neural Information Processing Systems 18*. MIT Press, 2006.

[3] L. Xu, M. I. Jordan, and G. E. Hinton. An alternative model for mixtures of experts. In *Advances in Neural Information Processing Systems 7*. MIT Press, 1995.

[4] N. Ueda and Z. Ghahramani. Bayesian model search for mixture models based on optimizing variational bounds. *Neural Networks*, 15(10):1223–1241, 2002.

[5] C. E. Rasmussen and C. K. I. Williams. *Gaussian Processes for Machine Learning*. MIT Press, 2006.

[6] A. J. Smola and P. Bartlett. Sparse greedy Gaussian process regression. In *Advances in Neural Information Processing Systems 13*. MIT Press, 2001.

[7] M. Seeger, C. K. I. Williams, and N. D. Lawrence. Fast forward selection to speed up sparse Gaussian process regression. In *Workshop on Artificial Intelligence and Statistics 9*, 2003.

[8] S. S. Keerthi and W. Chu. A matching pursuit approach to sparse Gaussian process regression. In *Advances in Neural Information Processing Systems 18*. MIT Press, 2006.

[9] E. Snelson and Z. Ghahramani. Sparse Gaussian processes using pseudo-inputs. In *Advances in Neural Information Processing Systems 18*. MIT Press, 2006.

[10] R. A. Jacobs, M. I. Jordan, S. J. Nowlan, and G. E. Hinton. Adaptive mixture of local experts. *Neural computation*, 3:79–87, 1991.

[11] S. Waterhouse. *Classification and regression using mixtures of experts*. PhD Theis, Department of Engineering, Cambridge University, 1997.

[12] C. M. Bishop and M. Svensén. Bayesian hierarchical mixtures of experts. In *Proc. Uncertainty in Artificial Intelligence*, 2003.

[13] V. Tresp. Mixtures of Gaussian processes. In *Advances in Neural Information Processing Systems 13*. MIT Press, 2001.

[14] B. W. Silverman. Some aspects of the spline smoothing approach to non-parametric regression curve fitting. *J. Royal. Stat. Society. B*, 47(1):1–52, 1985.

[15] C. E. Rasmussen. The infinite Gaussian mixture model. In *Advances in Neural Information Processing Systems 12*. MIT Press, 2000.

[16] L. Csató and M. Opper. Sparse on-line Gaussian processes. *Neural Computation*, 14(3):641–668, 2002.

